# Nonparametric Bayesian Inverse Reinforcement Learning for Multiple Reward Functions

**Jaedeug Choi and Kee-Eung Kim**
Department of Computer Science
Korea Advanced Institute of Science and Technology
Daejeon 305-701, Korea
jdchoi@ai.kaist.ac.kr, kekim@cs.kaist.ac.kr

## Abstract

We present a nonparametric Bayesian approach to inverse reinforcement learning (IRL) for multiple reward functions. Most previous IRL algorithms assume that the behaviour data is obtained from an agent who is optimizing a single reward function, but this assumption is hard to guarantee in practice. Our approach is based on integrating the Dirichlet process mixture model into Bayesian IRL. We provide an efficient Metropolis-Hastings sampling algorithm utilizing the gradient of the posterior to estimate the underlying reward functions, and demonstrate that our approach outperforms previous ones via experiments on a number of problem domains.

## 1  Introduction

Inverse reinforcement learning (IRL) aims to find the agent's underlying reward function given the behaviour data and the model of environment [1]. IRL algorithms often assume that the behaviour data is from an agent who behaves optimally without mistakes with respect to a single reward function. From the Markov decision process (MDP) perspective, the IRL can be defined as the problem of finding the reward function given the trajectory data of an optimal policy, consisting of state-action histories. Under this assumption, a number of studies on IRL have appeared in the literature [2, 3, 4, 5]. In addition, IRL has been applied to various practical problems that includes inferring taxi drivers' route preferences from their GPS data [6], estimating patients' preferences to determine the optimal timing of living-donor liver transplants [7], and implementing simulated users to assess the quality of dialogue management systems [8].

In practice, the behaviour data is often gathered collectively from multiple agents whose reward functions are potentially different from each other. The amount of data generated from a single agent may be severely limited, and hence we may suffer from the sparsity of data if we try to infer the reward function individually. Moreover, even when we have enough data from a single agent, the reward function may change depending on the situation.

However, most of the previous IRL algorithms assume that the behaviour data is generated by a single agent optimizing a fixed reward function, although there are a few exceptions that address IRL for multiple reward functions. Dimitrakakis and Rothkopf [9] proposed a multi-task learning approach, generalizing the Bayesian approach to IRL [4]. In this work, the reward functions are individually estimated for each trajectory, which are assumed to share a common prior. Other than the common prior assumption, there is no effort to group trajectories that are likely to be generated from the same or similar reward functions. On the other hand, Babeş-Vroman *et al.* [10] took a more direct approach that combines EM clustering with IRL algorithm. The behaviour data are clustered

based on the inferred reward functions, where the reward functions are defined per cluster. However, the number of clusters (hence the number of reward functions) has to be specified as a parameter in order to use the approach.

In this paper, we present a nonparametric Bayesian approach using the Dirichlet process mixture model in order to address the IRL problem with multiple reward functions. We develop an efficient Metropolis-Hastings (MH) sampler utilizing the gradient of the reward function posterior to infer reward functions from the behaviour data. In addition, after completing IRL on the behaviour data, we can efficiently estimate the reward function for a new trajectory by computing the mean of the reward function posterior given the pre-learned results.

## 2 Preliminaries

We assume that the environment is modeled as an MDP $\langle S, A, T, R, \gamma, b_0 \rangle$ where: $S$ is the finite set of states; $A$ is the finite set of actions; $T(s, a, s')$ is the state transition probability of changing to state $s'$ from state $s$ when action $a$ is taken; $R(s, a)$ is the immediate reward of executing action $a$ in state $s$; $\gamma \in [0, 1)$ is the discount factor; $b_0(s)$ denotes the probability of starting in state $s$. For notational convenience, we use the vector $\boldsymbol{r} = [r_1, \ldots, r_D]$ to denote the reward function.[1]

A policy is a mapping $\pi : S \rightarrow A$. The value of policy $\pi$ is the expected discounted return of executing the policy, defined as $V^\pi = \mathbb{E}\left[\sum_{t=0}^{\infty} \gamma^t R(s_t, a_t) | b_0, \pi\right]$. The value function of policy $\pi$ for each state $s$ is computed by $V^\pi(s) = R(s, \pi(s)) + \gamma \sum_{s' \in S} T(s, \pi(s), s') V^\pi(s')$ so that the value is calculated by $V^\pi = \sum_{s \in S} b_0(s) V^\pi(s)$. Similarly, the $Q$-function is defined as $Q^\pi(s, a) = R(s, a) + \gamma \sum_{s' \in S} T(s, a, s') V^\pi(s')$. Given an MDP, the agent's objective is to execute an optimal policy $\pi^*$ that maximizes the value function for all the states, which should satisfy the Bellman optimality equation: $V^*(s) = \max_{a \in A} \left[R(s, a) + \gamma \sum_{s' \in S} T(s, a, s') V^*(s')\right]$.

We assume that the agent's behavior data is generated by executing an optimal policy with some unknown reward function(s) $R$, given as the set $\mathcal{X}$ of $M$ trajectories where the $m$-th trajectory is an $H$-step sequence of state-action pairs: $\mathcal{X}_m = \{(s_{m,1}, a_{m,1}), (s_{m,2}, a_{m,2}), \ldots, (s_{m,H}, a_{m,H})\}$.[2]

### 2.1 Bayesian Inverse Reinforcement Learning (BIRL)

Ramachandran and Amir [4] proposed a Bayesian approach to IRL with the assumption that the behaviour data is generated from a single reward function. The prior encodes the the reward function preference and the likelihood measures the compatibility of the reward function with the data.

Assuming that the reward function entries are independently distributed, the prior is defined as $P(\boldsymbol{r}) = \prod_{d=1}^{D} P(r_d)$. We can use various distributions for the reward prior. For instance, the uniform distribution can be used if we have no knowledge or preference on rewards other than its range, and the normal or Laplace distributions can be used if we prefer rewards to be close to some specific values. The Beta distribution can also be used if we treat rewards as the parameter of the Bernoulli distribution, *i.e.* $P(\xi_d = 1) = r_d$ with auxiliary binary random variable $\xi_d$ [11].

The likelihood is defined as an independent exponential distribution, analogous to the softmax distribution over actions:

$$P(\mathcal{X}|\boldsymbol{r}, \eta) = \prod_{m=1}^{M} \prod_{h=1}^{H} P(a_{m,h}|s_{m,h}\boldsymbol{r}, \eta) = \prod_{m=1}^{M} \prod_{h=1}^{H} \frac{\exp(\eta Q^*(s_{m,h}, a_{m,h}; \boldsymbol{r}))}{\sum_{a'} \exp(\eta Q^*(s_{m,h}, a'; \boldsymbol{r}))} \quad (1)$$

where $\eta$ is the confidence parameter of choosing optimal actions and $Q^*(\cdot, \cdot; \boldsymbol{r})$ denotes the optimal $Q$-function computed using reward function $\boldsymbol{r}$.

For the sake of exposition, we assume that the reward function entries are independently and normally distributed with mean $\mu$ and variance $\sigma^2$ so that the prior is defined as $P(\boldsymbol{r}|\mu, \sigma) = \prod_{d=1}^{D} \mathcal{N}(r_d; \mu, \sigma)$, but our approach to be presented in later sections can be generalized to use many other distributions for the prior. The posterior over the reward functions is then formulated by

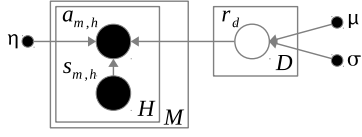

Figure 1: Graphical model for BIRL.

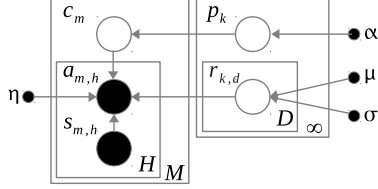

Figure 2: Graphical model for DPM-BIRL.

**Algorithm 1:** MH algorithm for DPM-BIRL

Initialize $\boldsymbol{c}$ and $\{\boldsymbol{r}_k\}_{k=1}^K$
**for** $t = 1$ **to** *MaxIter* **do**
&emsp;**for** $m = 1$ **to** $M$ **do**
&emsp;&emsp;$c_m^* \sim P(c|\boldsymbol{c}_{-m}, \alpha)$
&emsp;&emsp;**if** $c_m^* \notin \boldsymbol{c}_{-m}$ **then** $\boldsymbol{r}_{c_m^*} \sim P(\boldsymbol{r}|\mu, \sigma)$
&emsp;&emsp;$\langle c_m, \boldsymbol{r}_{c_m} \rangle \leftarrow \langle c_m^*, \boldsymbol{r}_{c_m^*} \rangle$ with prob. of
&emsp;&emsp;$\min\{1, \frac{P(\mathcal{X}_m|\boldsymbol{r}_{c_m^*}, \eta)}{P(\mathcal{X}_m|\boldsymbol{r}_{c_m}, \eta)}\}$
&emsp;**for** $k = 1$ **to** $K$ **do**
&emsp;&emsp;$\epsilon \sim \mathcal{N}(0, 1)$
&emsp;&emsp;$\boldsymbol{r}_k^* \leftarrow \boldsymbol{r}_k + \frac{\tau^2}{2}\nabla \log f(\boldsymbol{r}_k) + \tau\epsilon$
&emsp;&emsp;$\boldsymbol{r}_k \leftarrow \boldsymbol{r}_k^*$ with prob. of $\min\{1, \frac{f(\boldsymbol{r}_k^*)g(\boldsymbol{r}_k^*, \boldsymbol{r}_k)}{f(\boldsymbol{r}_k)g(\boldsymbol{r}_k, \boldsymbol{r}_k^*)}\}$

Bayes rule as follows:

$$P(\boldsymbol{r}|\mathcal{X}, \eta, \mu, \sigma) \propto P(\mathcal{X}|\boldsymbol{r}, \eta)P(\boldsymbol{r}|\mu, \sigma). \tag{2}$$

We can infer the reward function from the model by computing the posterior mean using a Markov chain Monte Carlo (MCMC) algorithm [4] or the maximum-a-posteriori (MAP) estimates using a gradient method [12]. Fig. 1 shows the graphical model used in BIRL.

## 3 Nonparametric Bayesian IRL for Multiple Reward Functions

In this section, we present our approach to IRL for multiple reward functions. We assume that each trajectory in the behaviour data is generated by an agent with a fixed reward function. In other words, we assume that the reward function does not change within a trajectory. However, the whole trajectories are assumed be generated by one or more agents whose reward functions are distinct from each other. We do not assume any information regarding which trajectory is generated by which agent as well as the number of agents. Hence, the goal is to infer an unknown number of reward functions from the unlabeled behaviour data.

A naive approach to this problem setting would be solving $M$ separate and independent IRL problems by treating each trajectory as the sole behaviour data and employing one of the well-known IRL algorithms designed for a single reward function. We can then use an unsupervised learning method with the $M$ reward functions as data points. However, this approach would suffer from the sparsity of data, since each trajectory may not contain a sufficient amount of data to infer the reward function reliably, or the number of trajectories may not be enough for the unsupervised learning method to yield a meaningful result. Babeş-Vroman *et al.* [10] proposed an algorithm that combines EM clustering with IRL algorithm. It clusters trajectories and assumes that all the trajectories in a cluster are generated by a single reward function. However, as a consequence of using EM clustering, we need to specify the number of clusters (*i.e.* the number of distinct reward functions) as a parameter.

We take a nonparametric Bayesian approach to IRL using the Dirichlet process mixture model. Our approach has three main advantages. First, we do not need to specify the number of distinct reward functions due to the nonparametric nature of our model. Second, we can encode our preference or domain knowledge on the reward function into the prior since it is a Bayesian approach to IRL. Third, we can acquire rich information from the behaviour data such as the distribution over the reward functions.

### 3.1 Dirichlet Process Mixture Models

The Dirichlet process mixture (DPM) model [13] provides a nonparametric Bayesian framework for clustering using mixture models with a countably infinite number of mixture components. The prior of the mixing distribution is given by the Dirichlet process, which is a distribution over distributions

parameterized by base distribution $G_0$ and concentration parameter $\alpha$. The DPM model for a data $\{x_m\}_{m=1}^M$ using a set of latent parameters $\{\theta_m\}_{m=1}^M$ can be defined as:

$$G|\alpha, G_0 \sim DP(\alpha, G_0),$$
$$\theta_m|G \sim G$$
$$x_m|\theta_m \sim F(\theta_m)$$

where $G$ is the prior used to draw each $\theta_m$ and $F(\theta_m)$ is the parameterized distribution for data $x_m$. This is equivalent to the following form with $K \to \infty$:

$$\boldsymbol{p}|\alpha \sim \text{Dirichlet}(\alpha/K, \dots, \alpha/K)$$
$$c_m|\boldsymbol{p} \sim \text{Multinomial}(p_1, \dots, p_K)$$
$$\phi_k \sim G_0$$
$$x_m|c_m, \boldsymbol{\phi} \sim F(\phi_{c_m}) \tag{3}$$

where $\boldsymbol{p} = \{p_k\}_{k=1}^K$ is the mixing proportion for the latent classes, $c_m \in \{1, \dots, K\}$ is the class assignment of $x_m$ so that $c_m = k$ when $x_m$ is assigned to class $k$, $\phi_k$ is the parameter of the data distribution for class $k$, and $\boldsymbol{\phi} = \{\phi_k\}_{k=1}^K$.

## 3.2 DPM-BIRL for Multiple Reward Functions

We address the IRL for multiple reward functions by extending BIRL with the DPM model. We place a Dirichlet process prior on the reward functions $\boldsymbol{r}_k$. The base distribution $G_0$ is defined as the reward function prior, *i.e.* the product of the normal distribution for each reward entry $\prod_{d=1}^D \mathcal{N}(r_{k,d}; \mu, \sigma)$. The cluster assignment $c_m = k$ indicates that the trajectory $\mathcal{X}_m$ belongs to the cluster $k$, which represents that the trajectory is generated by the agent with the reward function $\boldsymbol{r}_k$. We can thus regard the behavior data $\mathcal{X} = \{\mathcal{X}_1, \dots, \mathcal{X}_M\}$ as being drawn from the following generative process:

1. The cluster assignment $c_m$ is drawn by the first two equations in Eqn. (3).
2. The reward function $\boldsymbol{r}_k$ is drawn from $\prod_{d=1}^D \mathcal{N}(r_{k,d}; \mu, \sigma)$.
3. The trajectory $\mathcal{X}_m$ is drawn from $P(\mathcal{X}_m|\boldsymbol{r}_{c_m}, \eta)$ in Eqn. (1).

Fig. 2 shows the graphical model of DPM-BIRL. The joint posterior of the cluster assignment $\boldsymbol{c} = \{c_m\}_{m=1}^M$ and the set of reward functions $\{\boldsymbol{r}_k\}_{k=1}^K$ is defined as:

$$P(\boldsymbol{c}, \{\boldsymbol{r}_k\}_{k=1}^K | \mathcal{X}, \eta, \mu, \sigma, \alpha) = P(\boldsymbol{c}|\alpha) \prod_{k=1}^K P(\boldsymbol{r}_k | \mathcal{X}_{\boldsymbol{c}(k)}, \eta, \mu, \sigma) \tag{4}$$

where $\mathcal{X}_{\boldsymbol{c}(k)} = \{\mathcal{X}_m | c_m = k \text{ for } m = 1, \dots, M\}$ and $P(\boldsymbol{r}_k | \mathcal{X}, \eta, \mu, \sigma)$ are taken from Eqn. (2).

The inference in DPM-BIRL can be done using the Metropolis-Hastings (MH) algorithm that samples each hidden variable in turn. First, note that we can safely assume that there are $K$ distinct values of $c_m$'s so that $c_m \in \{1, \dots, K\}$ without loss of generality. The conditional distribution to sample $c_m$ for the MH update can be defined as

$$P(c_m | \boldsymbol{c}_{-m}, \{\boldsymbol{r}_k\}_{k=1}^K, \mathcal{X}, \eta, \alpha) \propto P(\mathcal{X}_m | \boldsymbol{r}_{c_m}, \eta) P(c_m | \boldsymbol{c}_{-m}, \alpha)$$

$$P(c_m | \boldsymbol{c}_{-m}, \alpha) \propto \begin{cases} n_{-m, c_j}, & \text{if } c_m = c_j \text{ for some } j \\ \alpha, & \text{if } c_m \neq c_j \text{ for all } j \end{cases} \tag{5}$$

where $\boldsymbol{c}_{-m} = \{c_i | i \neq m \text{ for } i = 1, \dots, M\}$, $P(\mathcal{X}_m | \boldsymbol{r}_{c_m}, \eta)$ is the likelihood defined in Eqn. (1), and $n_{-m, c_j} = |\{c_i = c_j | i \neq m \text{ for } i = 1, \dots, M\}|$ is the number of trajectories, excluding $\mathcal{X}_m$, assigned to the cluster $c_j$. Note that if the sampled $c_m \neq c_j$ for all $j$ then $\mathcal{X}_m$ is assigned to a new cluster. The conditional distribution to sample $\boldsymbol{r}_k$ for the MH update is defined as

$$P(\boldsymbol{r}_k | \boldsymbol{c}, \boldsymbol{r}_{-k}, \mathcal{X}, \eta, \mu, \sigma) \propto P(\mathcal{X}_{\boldsymbol{c}(k)} | \boldsymbol{r}_k, \eta) P(\boldsymbol{r}_k | \mu, \sigma)$$

where $P(\mathcal{X}_{\boldsymbol{c}(k)} | \boldsymbol{r}_k, \eta)$ is again the likelihood defined in Eqn. (1) and $P(\boldsymbol{r}_k | \mu, \sigma) = \prod_{d=1}^D \mathcal{N}(r_{k,d}; \mu, \sigma)$.

In Alg. 1, we present the MH algorithm for DPM-BIRL that uses the above MH updates. The algorithm consists of two steps. The first step updates the cluster assignment $\boldsymbol{c}$. We sample new

assignment $c_m^*$ from Eqn. (5). If $c_m^*$ is not in $\boldsymbol{c}_{-m}$, *i.e.*, $c_m^* \neq c_j$ for all $j$, we draw new reward function $\boldsymbol{r}_{c_m^*}$ from the reward prior $P(\boldsymbol{r}|\mu, \sigma)$. We then set $c_m = c_m^*$ with the acceptance probability of $\min\{1, \frac{P(\mathcal{X}_m|\boldsymbol{r}_{c_m^*}, \eta)}{P(\mathcal{X}_m|\boldsymbol{r}_{c_m}, \eta)}\}$, since we are using a non-conjugate prior [13]. The second step updates the reward functions $\{\boldsymbol{r}_k\}_{k=1}^K$. We sample a new reward function $\boldsymbol{r}_k^*$ using the equation

$$\boldsymbol{r}_k^* = \boldsymbol{r}_k + \frac{\tau^2}{2}\nabla \log f(\boldsymbol{r}_k) + \tau\epsilon$$

where $\epsilon$ is a sample from the standard normal distribution $\mathcal{N}(0, 1)$, $\tau$ is a non-negative scalar for the scaling parameter, and $f(\boldsymbol{r}_k)$ is the target distribution of the MH update $P(\mathcal{X}_{c(k)}|\boldsymbol{r}_k, \eta)P(\boldsymbol{r}_k|\mu, \sigma)$ which is the unnormalized posterior of the reward function $\boldsymbol{r}_k$. We then set $\boldsymbol{r}_k = \boldsymbol{r}_k^*$ with the acceptance probability of $\min\{1, \frac{f(\boldsymbol{r}_k^*)g(\boldsymbol{r}_k^*, \boldsymbol{r}_k)}{f(\boldsymbol{r}_k)g(\boldsymbol{r}_k, \boldsymbol{r}_k^*)}\}$ where

$$g(\boldsymbol{x}, \boldsymbol{y}) = \frac{1}{(2\pi\tau^2)^{D/2}} \exp\left(-\frac{1}{2\tau^2}||\boldsymbol{x} - \boldsymbol{y} - \frac{1}{2}\tau^2\nabla\log f(\boldsymbol{x})||_2^2\right).$$

This step is motivated by the Langevin algorithm [14] which exploits local information (*i.e.* gradient) of $f$ in order to efficiently move towards the high probability region. This algorithm is known to be more efficient than random walk MH algorithms. We can compute the gradient of $f$ using the results of Choi and Kim [12].

## 3.3 Information Transfer to a New Trajectory

Suppose that we would like to infer the reward function of a new trajectory after we finish IRL on the behaviour data consisting of $M$ trajectories. A naive approach would be running IRL from scratch using all of the $M + 1$ trajectories. However, it would be more desirable to transfer the relevant information from the pre-computed IRL results. In order to do so, Babeş-Vroman *et al.* [10] use the weighted average of cluster reward functions assuming that the new trajectory is generated from the same population of the behaviour data. Note that we can relax this assumption and allow the new trajectory generated by a novel reward function, as a direct result of using DPM model.

Given the cluster assignment $\boldsymbol{c}$ and the reward functions $\{\boldsymbol{r}_k\}_{k=1}^K$ computed from the behaviour data, the conditional prior of the reward function $\boldsymbol{r}$ for the new trajectory can be defined as:

$$P(\boldsymbol{r}|\boldsymbol{c}, \{\boldsymbol{r}_k\}_{k=1}^K, \mu, \sigma, \alpha) = \frac{\alpha}{\alpha+M}P(\boldsymbol{r}|\mu, \sigma) + \frac{1}{\alpha+M}\sum_{k=1}^K n_k\delta(\boldsymbol{r} - \boldsymbol{r}_k) \quad (6)$$

where $n_k = |\{\mathcal{X}_m|c_m = k \text{ for } m = 1, \dots, M\}|$ is the number of trajectories assigned to cluster $k$ and $\delta(x)$ is the Dirac delta function. Running Alg. 1 on the behaviour data $\mathcal{X}$, we already have a set of $N$ samples $\{\boldsymbol{c}^{(n)}, \{\boldsymbol{r}_k^{(n)}\}_{k=1}^{K^{(n)}}\}_{n=1}^N$ drawn from the joint posterior. The conditional posterior of $\boldsymbol{r}$ for the new trajectory $\mathcal{X}_{\text{new}}$ is then:

$$P(\boldsymbol{r}|\mathcal{X}_{\text{new}}, \mathcal{X}, \Theta) \propto P(\mathcal{X}_{\text{new}}|\boldsymbol{r}, \eta)P(\boldsymbol{r}|\mathcal{X}, \Theta)$$

$$= P(\mathcal{X}_{\text{new}}|\boldsymbol{r}, \eta)\int P(\boldsymbol{r}|\boldsymbol{c}, \{\boldsymbol{r}_k\}_{k=1}^K, \mu, \sigma, \alpha)\mathrm{d}P(\boldsymbol{c}, \{\boldsymbol{r}_k\}_{k=1}^K|\mathcal{X}, \Theta)$$

$$\approx P(\mathcal{X}_{\text{new}}|\boldsymbol{r}, \eta)\frac{1}{N}\sum_{n=1}^N P(\boldsymbol{r}|\{\boldsymbol{c}^{(n)}, \{\boldsymbol{r}_k^{(n)}\}_{k=1}^{K^{(n)}}\}_{n=1}^N, \mu, \sigma, \alpha)$$

$$= P(\mathcal{X}_{\text{new}}|\boldsymbol{r}, \eta)\left[\frac{\alpha}{\alpha+M}P(\boldsymbol{r}|\mu, \sigma) + \frac{1}{\alpha+M}\sum_{n=1}^N\sum_{k=1}^{K^{(n)}}\frac{n_k^{(n)}}{N}\delta(\boldsymbol{r} - \boldsymbol{r}_k^{(n)})\right]$$

where $\Theta = \{\eta, \mu, \sigma, \alpha\}$.

We can then re-draw samples of $\boldsymbol{r}$ using the approximated posterior and take the sample average as the inferred reward function. However, we present a more efficient way of calculating the posterior mean of $\boldsymbol{r}$ without re-drawing the samples. Note that Eqn. (6) is a mixture of a continuous distribution $P(\boldsymbol{r}|\mu, \sigma)$ with a number of point mass distributions on $\{\boldsymbol{r}_k\}_{k=1}^K$. If we approximate the continuous one by a point mass distribution, *i.e.*, $P(\boldsymbol{r}|\mu, \sigma) \approx \delta(\hat{\boldsymbol{r}})$, the posterior mean is analytically computable using the above approximation:

$$\mathbb{E}[\boldsymbol{r}|\mathcal{X}_{\text{new}}, \mathcal{X}, \Theta] = \int \boldsymbol{r}\mathrm{d}P(\boldsymbol{r}|\mathcal{X}_{\text{new}}, \mathcal{X}, \Theta)$$

$$\approx \frac{1}{Z}\left[\alpha P(\mathcal{X}_{\text{new}}|\hat{\boldsymbol{r}}, \eta)\hat{\boldsymbol{r}} + \sum_{n=1}^N\sum_{k=1}^{K^{(n)}}\frac{n_k^{(n)}}{N}P(\mathcal{X}_{\text{new}}|\boldsymbol{r}_k^{(n)}, \eta)\boldsymbol{r}_k^{(n)}\right] \quad (7)$$

where $Z$ is the normalizing constant. We choose $\hat{\boldsymbol{r}} = \operatorname{argmax}_{\boldsymbol{r}} P(\mathcal{X}_{\text{new}}|\boldsymbol{r}, \eta)P(\boldsymbol{r}|\mu, \sigma)$, which is the MAP estimate of the reward function for the new trajectory $\mathcal{X}_{\text{new}}$ only, ignoring the previous behaviour data $\mathcal{X}$.

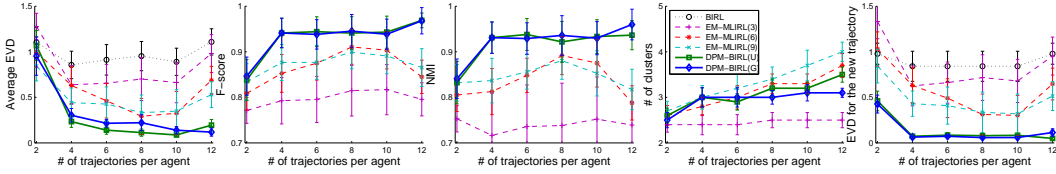

Figure 3: Results with increasing number of trajectories per agent in the gridworld problem. DPM-BIRL uses the uniform (U) and the standard normal (N) priors.

## 4 Experimental Results

We compared the performance of DPM-BIRL to the EM-MLIRL algorithm [10] and the baseline algorithm which runs BIRL separately on each trajectory. The experiments consisted of two tasks: The first task was finding multiple reward functions from the behaviour data with a number of trajectories. The second task was inferring the reward function underlying a new trajectory, while exploiting the results learned in the first task.

The performance of each algorithm was evaluated by the expected value difference (EVD) $|V^*(r^A) - V^{\pi^*(r^L)}(r^A)|$ where $r^A$ is the agent's ground truth reward function, $r^L$ is the learned reward function, $\pi^*(r)$ is the optimal policy induced by reward function $r$, and $V^\pi(r)$ is the value of policy $\pi$ measured using $r$. The EVD thus measures the performance difference between the agent's optimal policy and the optimal policy induced by the learned reward function. In the first task, we evaluated the EVD for the true and learned reward functions of each trajectory and computed the average EVD over the trajectories in the behaviour data. In the second task, we evaluated the EVD for the new trajectory. The clustering quality on the behaviour data was evaluated by F-score and normalized mutual information (NMI).

In all the experiments, we assumed that the reward function was linearly parameterized such that $R(s,a) = \sum_{d=1}^{D} r_d \phi_d(s,a)$ with feature functions $\phi_d : S \times A \to \mathbb{R}$, hence $r = [r_1, \ldots, r_D]$.

### 4.1 Gridworld Problem

In order to extensively evaluate our approach, we first performed experiments on a small toy domain, $8 \times 8$ gridworld, where each of the 64 cells corresponds to the state. The agent can move north, south, east, or west, but with probability of 0.2, it fails and moves in a random direction. The initial state is randomly chosen from the states. The grid is partitioned into non-overlapping regions of size $2 \times 2$, and the feature function is defined by a binary indicator function for each region. Random instances of IRL with three reward functions were generated as follows: each element of $r$ was sampled to have a non-zero value with probability of 0.2 and the value is drawn from the uniform distribution between -1 and 1. We obtained the trajectories of 40 time steps and measured the performance as we increased the number of trajectories per reward function.

Fig. 3 shows the averages and standard errors of the performance results over 10 problem instances. The left four panels in the figure present the results for the first task of learning multiple reward functions from the behaviour data. When the size of the behaviour data is small, the clustering performances of both DPM-BIRL and EM-MLIRL were not good enough due to the sparsity of data, hence their EVD results were similar to that of the baseline algorithm that independently runs BIRL on each trajectory. However, as we increased the size of the data, both DPM-BIRL and EM-MLIRL achieved better EVD results than the baseline since they could utilize more information by grouping the trajectories to infer the reward functions. As for EM-MLIRL, we set the parameter $K$ used for the maximum number of clusters to 3 (ground truth), 6 (2x), and 9 (3x). DPM-BIRL achieved significantly better results than EM-MLIRL with all of the parameter settings, in terms of EVD and clustering quality. The rightmost panel in the figure present the results for the second task of inferring the reward function for a new trajectory. DPM-BIRL clearly outperformed EM-MLIRL since it exploits the rich information from the reward function posterior. The relatively large error bars of the EM-MLIRL results are due to the local convergence inherent to EM clustering.

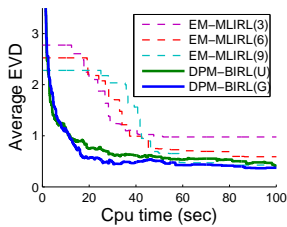

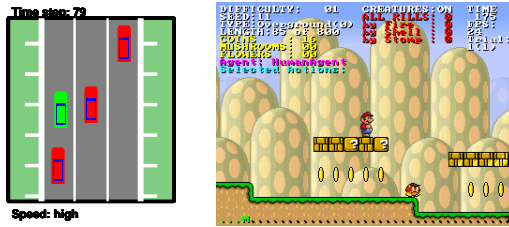

Figure 4: CPU timing results in the gridworld problem.

Figure 5: Screenshots of *Simulated-highway* problem (left) and *Mario Bros* (right).

Table 1: Results in *Simulated-highway* problem.

|  | Average EVD | F-score | NMI | # of clusters | EVD for $\mathcal{X}_{\text{new}}$ |
|---|---|---|---|---|---|
| BIRL | 0.52±0.05 | n.a. | n.a. | n.a. | 0.41±0.00 |
| EM-MLIRL(3) | 4.53±0.96 | 0.80±0.05 | 0.74±0.09 | 2.20±0.20 | 4.14±0.88 |
| EM-MLIRL(6) | 0.89±0.57 | 0.96±0.02 | 0.96±0.03 | 3.10±0.18 | 0.82±0.53 |
| DPM-BIRL(U) | 0.35±0.04 | 0.98±0.01 | 0.97±0.01 | 3.30±0.15 | 0.32±0.04 |
| DPM-BIRL(N) | 0.36±0.05 | 0.99±0.01 | 0.99±0.01 | 3.10±0.10 | 0.30±0.04 |

Fig. 4 compares the average CPU timing results of DPM-BIRL and EM-MLIRL with 10 trajectories per reward function. DPM-BIRL using Alg. 1 took much shorter time to converge than EM-MLIRL. This is mainly due to the fact that, whereas EM-MLIRL performs full single-reward IRL multiple times in each iteration, DPM-BIRL takes a sample from the posterior leveraging the gradient that does not involve a full IRL.

## 4.2   Simulated-highway Problem

The second set of experiments was conducted in *Simulated-highway* problem [15] where the agent drives on a three lane road. The left panel in Fig. 5 shows a screenshot of the problem. The agent can move one lane left or right and drive at speeds 2 through 3, but it fails to change the lane with probability of 0.2 and 0.4 respectively in speed 2 and 3. All the other cars on the road constantly drive at speed 1 and do not change the lane. The reward function is defined by using 6 binary feature functions: one function for indicating the agent's collision with other cars, 3 functions for indicating the agent's current lane, 2 functions for indicating the agent's current speed. We generated three agents having different driving styles. The first one prefers driving at speed 3 in the left-most lane and avoiding collisions. The second one prefers driving at speed 3 in the right-most lane and avoiding collisions. The third one prefers driving at speed 2 and colliding with other cars. We prepared 3 trajectories of 40 time steps per driver agent for the first task and 20 trajectories of 40 time steps yielded by a driver randomly chosen among the three for the second task.

Tbl. 1 presents the averages and standard errors of the results over 10 sets of the behaviour data. DPM-BIRL significantly outperformed the others while EM-MLIRL suffered from the convergence to a local optimum.

## 4.3   Mario Bros.

For the third set of experiments, we used the open source simulator of the game *Mario Bros*, which is a challenging problem due to its huge state space. The right panel in Fig. 5 is a screenshot of the game. Mario can move left, move right, or jump. Mario's goal is to reach the end of the level by traversing from left to right while collecting coins and avoiding or killing enemies. We used 8 binary feature functions, each being an indicator for: Mario successfully reaching the end of the level; Mario getting killed; Mario killing an enemy; Mario collecting a coin; Mario receiving damage by an enemy; existence of a wall preventing Mario from moving in the current direction; Mario moving to the right; Mario moving to the left. We collected the behaviour data from 4 players: The expert player is good at both collecting coins and killing enemies. The coin collector likes to collect coins but avoids killing enemies. The enemy killer likes to kill enemies but avoids collecting coins. The

Table 2: Cluster assignments in *Mario Bros*.

| $c$ | Expert player | | | Coin collector | | | Enemy killer | | | Speedy Gonzales | | |
|---|---|---|---|---|---|---|---|---|---|---|---|---|
| DPM-BIRL | 1 | 1 | 1 | 1 | 2 | 2 | 3 | 3 | 4 | 5 | 5 | 5 |
| EM-MLIRL(4) | 1 | 1 | 1 | 1 | 1 | 2 | 2 | 2 | 1 | 3 | 3 | 3 |
| EM-MLIRL(8) | 1 | 1 | 1 | 1 | 2 | 2 | 3 | 3 | 1 | 3 | 3 | 3 |

Table 3: Results of DPM-BIRL in *Mario Bros*.

| $k$ from DPM-BIRL | Reward function entry ($r_{k,d}$) | | | | | Average feature counts | | | | |
|---|---|---|---|---|---|---|---|---|---|---|
| | 1 | 2 | 3 | 4 | 5 | 1 | 2 | 3 | 4 | 5 |
| $\phi_{\text{enemy-killed}}$ | 1.00 | -0.81 | 1.00 | 1.00 | -1.00 | 3.10 | 1.60 | 2.80 | 1.90 | 0.55 |
| $\phi_{\text{coin-collected}}$ | 1.00 | 1.00 | -1.00 | -0.42 | -1.00 | 21.60 | 21.55 | 7.55 | 7.85 | 6.75 |

speedy Gonzales avoids both collecting coins and killing enemies. All the players commonly try to reach the end of the level while acting according to their own preferences. The behaviour data consisted of 3 trajectories per player. Since only the simulator of the environment is available instead of the complete model, we used the relative entropy IRL [16] which is a model-free IRL algorithm.

Tbl. 2 presents the cluster assignment results. Each column represents each trajectory and the number denotes the cluster assignment $c_m$ of trajectory $\mathcal{X}_m$. For example, DPM-BIRL produced 5 clusters and trajectories $\mathcal{X}_1, \ldots, \mathcal{X}_4$ are assigned to the cluster 1 representing the expert player. EM-MLIRL failed to group the trajectories that align well with the players, even though we restarted it 100 times in order to mitigate the convergence to bad local optima. On the other hand, DPM-BIRL was incorrect on only one trajectory, assigning a coin collector's trajectory to the expert player cluster. Tbl. 3 presents the reward function entries ($r_{k,d}$) learned from DPM-BIRL and the average feature counts acquired by the players with the learned reward functions. For the sake of brevity, we present only two important features ($d$=enemy-killed, coin-collected) that determine the playing style. To compute each player's feature counts, we executed an $n$-step lookahead policy yielded by each reward function $\boldsymbol{r}_k$ on the simulator in 20 randomly chosen levels. The reward function entries align well with each playing style. For example, the cluster 2 represents the coin collector, and its reward function entry for killing an enemy is negative but that for collecting a coin is positive.

As a demonstration, we implemented a small piece of software that visualizes the posterior probability of a gamer's behavior belonging to one of the clusters including a new one. A demo video is provided as supplementary material.

# 5 Conclusion

We proposed a nonparametric Bayesian approach to IRL for multiple reward functions using the Dirichlet process mixture model, which extends the previous Bayesian approach to IRL assuming a single reward function. We can learn an appropriate number of reward functions from the behavior data due to the nonparametric nature and facilitates incorporating domain knowledge on the reward function by utilizing a Bayesian approach. We presented an efficient Metropolis-Hastings sampling algorithm that draws samples from the posterior of DPM-BIRL, leveraging the gradient of the posterior. We also provided an analytical way to compute the approximate posterior mean for the information transfer task. In addition, we showed that DPM-BIRL outperforms the previous approach in various problem domains.

**Acknowledgments**

This work was supported by National Research Foundation of Korea (Grant# 2012-007881), the Defense Acquisition Program Administration and Agency for Defense Development of Korea (Contract# UD080042AD), and the SW Computing R&D Program of KEIT (2011-10041313) funded by the Ministry of Knowledge Economy of Korea.

## Footnotes

[1]$D$ denotes the number of features. Note that we can assign individual reward values to every state-action pair by using $|S||A|$ indicator functions for features.

[2]Although we assume that all trajectories are of length $H$ for notational brevity, our formulation trivially extends to different lengths.

## References

[1] Stuart Russell. Learning agents for uncertain environments (extended abstract). In *Proceedings of COLT*, 1998.

[2] Andrew Y. Ng and Stuart Russell. Algorithms for inverse reinforcement learning. In *Proceedings of ICML*, 2000.

[3] Gergely Neu and Csaba Szepesvári. Apprenticeship learning using inverse reinforcement learning and gradient methods. In *Proceedings of UAI*, 2007.

[4] Deepak Ramachandran and Eyal Amir. Bayesian inverse reinforcement learning. In *Proceedings of IJCAI*, 2007.

[5] Brian D. Ziebart, Andrew L. Maas, J. Andrew Bagnell, and Anind K. Dey. Maximum entropy inverse reinforcement learning. In *Proceedings of AAAI*, 2008.

[6] Brian D. Ziebart, Andrew L. Maas, Anind K. Dey, and J. Andrew Bagnell. Navigate like a cabbie: probabilistic reasoning from observed context-aware behavior. In *Proceedings of the international conference on Ubiquitous computing*, 2008.

[7] Zeynep Erkin, Matthew D. Bailey, Lisa M. Maillart, Andrew J. Schaefer, and Mark S. Roberts. Eliciting patients' revealed preferences: An inverse Markov decision process approach. *Decision Analysis*, 7(4), 2010.

[8] Senthilkumar Chandramohan, Matthieu Geist, Fabrice Lefevre, and Olivier Pietquin. User simulation in dialogue systems using inverse reinforcement learning. In *Proceedings of Interspeech*, 2011.

[9] Christos Dimitrakakis and Constantin A. Rothkopf. Bayesian multitask inverse reinforcement learning. In *Proceedings of the European Workshop on Reinforcement Learning*, 2011.

[10] Monica Babeş-Vroman, Vukosi Marivate, Kaushik Subramanian, and Michael Littman. Apprenticeship learning about multiple intentions. In *Proceedings of ICML*, 2011.

[11] Peter Dayan and Geoffrey E. Hinton. Using expectation-maximization for reinforcement learning. *Neural Computation*, 9(2), 1997.

[12] Jaedeug Choi and Kee-Eung Kim. MAP inference for Bayesian inverse reinforcement learning. In *Proceedings of NIPS*, 2011.

[13] Radford M. Neal. Markov chain sampling methods for Dirichlet process mixture models. *Journal of Computational and Graphical Statistics*, 9(2), 2000.

[14] Gareth O. Roberts and Jeffrey S. Rosenthal. Optimal scaling of discrete approximations to langevin diffusions. *Journal of the Royal Statistical Society: Series B (Statistical Methodology)*, 60(1), 1998.

[15] Pieter Abbeel and Andrew Y. Ng. Apprenticeship learning via inverse reinforcement learning. In *Proceedings of ICML*, 2004.

[16] Abdeslam Boularias, Jens Kober, and Jan Peters. Relative entropy inverse reinforcement learning. In *Proceedings of AISTATS*, 2011.

